# Sensitivity analysis in HMMs
# with application to likelihood maximization

**Pierre-Arnaud Coquelin,**
Vekia, Lille, France
pacoquelin@vekia.fr

**Romain Deguest**[*]
Columbia University, New York City, NY 10027
rd2304@columbia.edu

**Rémi Munos**
INRIA Lille - Nord Europe, Sequel Project, France
remi.munos@inria.fr

## Abstract

This paper considers a sensitivity analysis in Hidden Markov Models with continuous state and observation spaces. We propose an Infinitesimal Perturbation Analysis (IPA) on the filtering distribution with respect to some parameters of the model. We describe a methodology for using any algorithm that estimates the filtering density, such as Sequential Monte Carlo methods, to design an algorithm that estimates its gradient. The resulting IPA estimator is proven to be asymptotically unbiased, consistent and has computational complexity linear in the number of particles.

We consider an application of this analysis to the problem of identifying unknown parameters of the model given a sequence of observations. We derive an IPA estimator for the gradient of the log-likelihood, which may be used in a gradient method for the purpose of likelihood maximization. We illustrate the method with several numerical experiments.

## 1 Introduction

We consider a parameterized hidden Markov model (HMM) defined on **continuous state and observation spaces**. The HMM is defined by a state process $(X_t)_{t \geq 0} \in X$ and an observation process $(Y_t)_{t \geq 1} \in Y$ that are parameterized by a continuous parameter $\theta = (\theta_1, \dots, \theta_d) \in \Theta$, where $\Theta$ is a compact subset of $\mathbb{R}^d$.

The **state process** is a Markov chain taking its values in a (measurable) state space $X$, with initial probability measure $\mu \in \mathcal{M}(X)$ (i.e. $X_0 \sim \mu$) and Markov transition kernel $K(\theta, x_t, dx_{t+1})$. We assume that we can sample this Markov chain using a transition function $F$ and independent random numbers, i.e. for all $t \geq 0$,

$$X_{t+1} = F(\theta, X_t, U_t), \text{ with } U_t \overset{i.i.d.}{\sim} \nu, \tag{1}$$

where $F : \Theta \times X \times U \to X$ and $(U, \sigma(U), \nu)$ is a probability space. In many practical situations $U = [0,1]^p$, $\nu$ is uniform, thus $U_t$ is a $p$-uple of uniform random numbers. For simplicity, we adopt the notations $F(\theta, x_{-1}, u) \triangleq F_\mu(\theta, u)$, where $F_\mu$ is the first transition function (i.e. $X_0 = F_\mu(\theta, U_{-1})$ with $U_{-1} \sim \nu$).

The **observation process** $(Y_t)_{t \geq 1}$ lies in a (measurable) space $Y$ and is linked with the state process by the conditional probability measure $\mathbb{P}(Y_t \in dy_t | X_t = x_t) = g(\theta, x_t, y_t) \, dy_t$, where $g : \Theta \times$

---

[*]also affiliated with CMAP, Ecole Polytechnique, France

$X \times Y \to [0, 1]$ is the marginal density function of $Y_t$ given $X_t$. We assume that observations are conditionally independent given the state.

Since the transition and observation processes are parameterized by the parameter $\theta$, the state $X_t$ and the observation $Y_t$ processes depend explicitly on $\theta$. For notation simplicity we will omit to write the dependence of $\theta$ (in $K$, $F$, $g$, $X_t$, $Y_t$, ...) when there is no possible ambiguity.

One of the main interest in HMMs is to recover the state at time $n$ given a sequence of past observations $(y_1, \ldots, y_n)$ (written $y_{1:n}$). The **filtering distribution** (or belief state)

$$\pi_n(dx_n) \triangleq \mathbb{P}(X_n \in dx_n | Y_{1:n} = y_{1:n})$$

is the distribution of $X_n$ conditioned on the information $y_{1:n}$. We define analogously the **predictive distribution**

$$\pi_{n+1|n}(dx_{n+1}) \triangleq \mathbb{P}(X_{n+1} \in dx_{n+1} | Y_{1:n} = y_{1:n}).$$

Our contribution is an **Infinitesimal Perturbation Analysis** (IPA) that estimates the gradient $\nabla \pi_n$ (where $\nabla$ refers to the derivative with respect to the parameter $\theta$) of the filtering distribution $\pi_n$. More precisely, we estimate $\nabla \pi_n(f)$ (where $\pi(f) \triangleq \int_X f(x) \pi(dx)$) for any integrable function $f$ under the filtering distribution $\pi_n$.

We also consider as application, the problem of parameter identification in HMMs which consists in estimating the (unknown) parameter $\theta^*$ of the model that has served to generate the sequence of observations. In a Maximum Likelihood (ML) approach, one searches for the parameter $\theta$ that maximizes the likelihood (or its logarithm) given the sequence of observations. The log-likelihood of parameter $\theta$ is defined by $l_n(\theta) \triangleq \log p_\theta(y_{1:n})$ where $p_\theta(y_{1:n}) \, dy_{1:n} \triangleq \mathbb{P}(Y_{1:n}(\theta) \in dy_{1:n})$. The Maximum Likelihood (ML) estimator $\hat{\theta}_n \triangleq \arg\max_{\theta \in \Theta} l_n(\theta)$ is asymptotically consistent (in the sense that $\hat{\theta}_n$ converges almost surely to the true parameter $\theta^*$ when $n \to \infty$ under some identifiably conditions and mild assumptions on the model, see Theorem 2 of [DM01]). Thus, using the ML approach, the parameter identification problem reduces to an optimization problem.

Our second contribution is a sensitivity analysis of the predictive distribution $\nabla \pi_{t+1|t}$, for $t < n$, which enables to estimate the gradient $\nabla l_n(\theta)$ of the log-likelihood function, which may be used in a (stochastic) gradient method for the purpose of optimizing the likelihood. The approach is numerically illustrated on two parameter identification problems (autoregressive model and a stochastic volatility model) and compared to other approaches (EM algorithm, the Kalman filter, and the Likelihood ratio approach) when these latter apply.

## 2  Links with other works

First, let us mention that we are interested in the continuous state case since numerous applications in signal processing, finance, robotics, or telecommunications naturally fit in this framework. In the general setting there exists no closed-form expression of the filtering distribution (unlike in finite spaces where the Viterbi algorithm may apply or in linear-Gaussian models where the Kalman filter can be used). Thus, in this paper, we will make use of the so-called **Sequential Monte Carlo** methods (SMC) (also known as Particle Filters) which are numerical tools that can be applied to a large class of models, see e.g. [DFG01]. For illustration, a challenging example in finance is the problem of parameter estimation in the stochastic volatility model, which is a non-linear non-Gaussian continuous space HMM parameterized by three continuous parameters (see e.g. [ME07]) which will be described in the experimental section.

A usual approach for parameter estimation consists in performing a maximum likelihood estimation (MLE), i.e. search for the most likely value of the parameter, given the observed data. For finite state space problems, the Expectation Maximization (EM) algorithm is a popular method for solving the MLE problem. However, in continuous space problems, see [CM05], the EM algorithm is difficult to use mainly because the Expectation part relies on the estimation of the posterior path measure which is intractable in many situations. The Maximization part may also be very complicated and time-consuming when the model does not belong to a linear or exponential family. An alternative method consists in using brute force optimization methods based on the evaluation of the likelihood such as grid-based or simulated annealing methods. These approaches, which can be seen as black-box optimization are not very efficient in high dimensional parameter spaces.

Another approach is to treat the parameter as part of the state variable and then compute the optimal filter (see [DFG01] and [Sto02]). In this case, the Bayesian posterior distribution of the parameter is a marginal of the optimal filter. It is well known that those methods are stable only under certain conditions, see [Pap07], and do not perform well in practice for a large number of time steps.

A last solution consists in using an optimization procedure based on the evaluation of the gradient of the log-likelihood function with respect to the parameter. These approaches have been studied in the field of continuous space HMMs e.g. in [DT03, FLM03, PDS05, Poy06]. The idea was to use a likelihood ratio approach (also called score method) to evaluate the gradient of the likelihood. This approach suffers from high variance of the estimator, in particular for problems with small noise in the dynamic. To tackle this issue, [PDS05] proposed to use a marginal particle filter instead of a simple path-based particle filter as Monte Carlo approximation method. This approach is efficient in terms of variance reduction but its computational complexity becomes quadratic in the number of particles instead of being linear, like in path-based particle methods.

The IPA approach proposed in this paper is an alternative gradient-based maximum likelihood approach. Compared with works on gradient approaches previously cited, the IPA provides usually a lower variance estimators than the likelihood ratio methods, and its numerical complexity is linear in the number of particles.

Other works related to ours are the so-called *tangent filter* approach described in [CGN01] for dynamics coming from a discretization of a diffusion process, and the *Finite-Difference* (FD) approach described in a different setting (i.e. policy gradient in Partially Observable Markov Decision Processes) in [CDM08]. A similar FD estimator could be designed in our setting too but the resulting FD estimator would be biased (like usual FD schemes) whereas the IPA estimator is not.

## 3  Sequential Monte Carlo methods (SMC)

Given a measurable test function $f : \mathcal{X} \to \mathbb{R}$, we have:

$$\pi_n(f) \triangleq \mathbb{E}[f(X_n)|Y_{1:n} = y_{1:n}] = \frac{\int f(x_n) \prod_{t=0}^n K(x_{t-1}, dx_t) G_t(x_t)}{\int \prod_{t=0}^n K(x_{t-1}, dx_t) G_t(x_t)} = \frac{\mathbb{E}[f(X_n) \prod_{t=0}^n G_t(X_t)]}{\mathbb{E}[\prod_{t=0}^n G_t(X_t)]}.$$
(2)

where we used the simplified notation: $G_t(x_t) \triangleq g(x_t, y_t)$ and $G_0(x_0) \triangleq 1$.

In general, it is impossible to write $\pi_n(f)$ analytically except for specific cases (such as linear/Gaussian with Kalman filtering). In this paper, we consider a numerical approximation of $\pi_n(f)$ based on a SMC method. But it should be mentioned that other methods (such as Extended Kalman filter, quantization methods, Markov Chain Monte Carlo methods) may be used as well to build the IPA estimator that we propose in the next section.

The basic SMC method, called Bootstrap Filter, see [DFG01] for details, approximates $\pi_n(f)$ by an empirical distribution $\pi_n^N(f) \triangleq \frac{1}{N} \sum_{i=1}^N f(x_n^i)$ made of $N$ particles $x_n^{1:N}$.

---

**Algorithm 1** Generic Sequential Monte Carlo

    **for** $t = 1$ **to** $n$ **do**

        **Sampling**: Sample $u_{t-1}^i \overset{iid}{\sim} \nu$ and set $\widetilde{x}_t^i = F(x_{t-1}^i, u_{t-1}^i), \forall i \in \{1, \ldots, N\}$. Then define the importance sampling weights $w_t^i = \frac{G_t(\widetilde{x}_t^i)}{\sum_{j=1}^N G_t(\widetilde{x}_t^j)}$,

        **Resampling**: Set $x_t^i = \widetilde{x}_t^{k_i}, \forall i \in \{1, \ldots, N\}$, where $k_{1:N}$ are indices selected from the weights $w_t^{1:N}$.

    **end for**

    RETURN: $\pi_n^N(f) = \frac{1}{N} \sum_{i=1}^N f(x_n^i)$

---

The sampling (or transition) step generates a successor particle population $\widetilde{x}_t^{1:N}$ according to the state dynamics from the previous population $x_{t-1}^{1:N}$. The importance sampling weights $w_t^{1:N}$ are evaluated, and the resampling (or selection) step resamples (with replacement) $N$ particles $x_t^{1:N}$ from the set $\widetilde{x}_t^{1:N}$ according to the weights $w_t^{1:N}$. Resampling is used to avoid the problem of degeneracy of the algorithm, i.e. that most of the weights decreases to zero. It consists in selecting new parti-

cle positions such as to preserve a consistency property (i.e. $\sum_{i=1}^{N} w_t^i \phi(\widetilde{x}_t^i) = \mathbb{E}[\frac{1}{N} \sum_{i=1}^{N} \phi(x_t^i)]$). The simplest version introduced in [GSS93] chooses the selection indices $k_t^{1:N}$ by an independent sampling from the set $\{1, \ldots, N\}$ according to a multinomial distribution with parameters $w_t^{1:N}$, i.e. $\mathbb{P}(k_t^i = j) = w_t^j$, for all $1 \le i \le N$. The idea is to replicate the particles in proportion to their weights. Many variants have been proposed in the literature, among which the stratified resampling method [Kit96] which is optimal in terms of variance minimization.

Convergence issues of $\pi_n^N(f)$ to $\pi_n(f)$ (e.g. Law of Large Numbers or Central Limit Theorems) are discussed in [Del04] or [DM08]. For our purpose we note that under mild conditions on $f$, $\pi_n^N(f)$ is an asymptotically unbiased (see [DMDP07] for the asymptotic expression of the bias) and consistent estimator of $\pi_n(f)$.

# 4 Infinitesimal Perturbation Analysis in HMMs

## 4.1 Sensitivity analysis of the filtering distribution

The following decomposition of the gradient of the filtering distribution $\pi_n$ applied to a function $f$:

$$\nabla[\pi_n(f)] = \nabla \left[ \frac{\mathbb{E}[f(X_n) \prod_{t=0}^{n} G_t(X_t)]}{\mathbb{E}[\prod_{t=0}^{n} G_t(X_t)]} \right] = \frac{\nabla \mathbb{E}[f(X_n) \prod_{t=0}^{n} G_t(X_t)]}{\mathbb{E}[\prod_{t=0}^{n} G_t(X_t)]} - \pi_n(f) \frac{\nabla \mathbb{E}[\prod_{t=0}^{n} G_t(X_t)]}{\mathbb{E}[\prod_{t=0}^{n} G_t(X_t)]} \tag{3}$$

shows that the problem of finding an estimator of $\nabla \pi_n(f)$ is reduced to the problem of finding an estimator of $\nabla \mathbb{E}[f(X_n) \prod_{t=0}^{n} G_t(X_t)]$. There are two dominant infinitesimal methods for estimating the gradient of an expectation in a Markov chain: the Infinitesimal Perturbation Analysis (IPA) method and the Score Function (SF) method (also called likelihood ratio method), see for instance [Gla91] and [Pfl96] for a detailed presentation of both methods. SF has been used in [DT03, FLM03] to estimate $\nabla \pi_n$. Although IPA is known for having a lower variance than SF in general, as far as we know, it has never been used in this context. This is therefore the object of this Section.

Under appropriate smoothness assumptions (see Proposition 1 below), the gradient of an expectation over a random variable $X$ is equal to an expectation involving the pair of random variables $(X, \nabla X)$

$$\nabla \mathbb{E}[f(X)] = \mathbb{E}[\nabla[f(X)]] = \mathbb{E}[f'(X)\nabla X],$$

(where $'$ refers to the derivative with respect to the state variable). Applying this property to estimate $\nabla \mathbb{E}[f(X_n) \prod_{t=0}^{n} G_t(X_t)]$, we deduce

$$\nabla \mathbb{E} \left[ f(X_n) \prod_{t=0}^{n} G_t(X_t) \right] = \mathbb{E} \left[ \nabla \left[ f(X_n) \prod_{t=0}^{n} G_t(X_t) \right] \right]$$

$$= \mathbb{E} \left[ \left( \nabla[f(X_n)] + f(X_n) \sum_{t=0}^{n} \frac{\nabla[G_t(X_t)]}{G_t(X_t)} \right) \prod_{t=0}^{n} G_t(X_t) \right]$$

$$= \mathbb{E} \left[ \left( f'(X_n)\nabla X_n + f(X_n) \sum_{t=0}^{n} \frac{G_t'(X_t)\nabla X_t + \nabla G_t(X_t)}{G_t(X_t)} \right) \prod_{t=0}^{n} G_t(X_t) \right]. \tag{4}$$

Now we define an augmented Markov chain $(X_t, Z_t, R_t)_{t \ge 0}$ by the following recursive relations (where $Z_t \triangleq \nabla X_t$)

$$\begin{cases} X_0 &= F_\mu(U_{-1}), \ U_{-1} \sim \nu \\ Z_0 &= \nabla F_\mu(U_{-1}), \\ R_0 &= 0, \end{cases} \quad \forall t \ge 0, \quad \begin{cases} X_{t+1} &= F(X_t, U_t), \text{ where } U_t \sim \nu \\ Z_{t+1} &= \nabla F(X_t, U_t) + F'(X_t, U_t)Z_t, \\ R_{t+1} &= R_t + \frac{G_{t+1}'(X_{t+1})Z_{t+1} + \nabla G_{t+1}(X_{t+1})}{G_{t+1}(X_{t+1})}, \end{cases}$$

By introducing this augmented Markov Chain in Equation (4) and using Equation (3) we can rewrite $\nabla \pi_n(f)$ as:

$$\begin{aligned} \nabla \pi_n(f) &= \frac{\mathbb{E}[(f'(X_n)Z_n + f(X_n)R_n) \prod_{t=0}^{n} G_t(X_t)]}{\mathbb{E}[\prod_{t=0}^{n} G_t(X_t)]} - \pi_n(f) \frac{\mathbb{E}[R_n \prod_{t=0}^{n} G_t(X_t)]}{\mathbb{E}[\prod_{t=0}^{n} G_t(X_t)]} \\ &= \frac{\mathbb{E}[(f'(X_n)Z_n + R_n(f(X_n) - \pi_n(f))) \prod_{t=0}^{n} G_t(X_t)]}{\mathbb{E}[\prod_{t=0}^{n} G_t(X_t)]}. \end{aligned} \tag{5}$$

We now state some sufficient conditions under which the previous derivations are sound.

**Proposition 1.** *Equation (5) is valid on $\Theta$ whenever the following conditions are satisfied:*

- *for all $\theta \in \Theta$, the path $\theta \mapsto (X_0, X_1, \cdots, X_n)(\theta)$ is almost surely (a.s.) differentiable,*

- *for all $\theta \in \Theta$, $f$ is a.s. continuously differentiable at $X_n(\theta)$, and for all $1 \leq t \leq n$, $G_t$ is a.s. continuously differentiable at $(\theta, X_t(\theta))$,*

- *$\theta \mapsto f(X_n(\theta))$ and for all $1 \leq t \leq n$, $\theta \mapsto G_t(\theta, X_t(\theta))$ are a.s. continuous and piecewise differentiable throughout $\Theta$,*

- *Let $D$ be the random subset of $\Theta$ at which $f(X_n(\theta))$ or one $G_t(\theta, X_t(\theta))$ fails to be differentiable. We require that $\mathbb{E}[\sup_{\theta \notin D} |f'(X_n) Z_n + R_n (f(X_n) - \pi_n(f))| \prod_{t=0}^{n} G_t(X_t)] < \infty$,*

The proof of this Proposition is a direct application of Theorem 1.2 from [Gla91]. We notice that requiring the a.s. differentiability of the path $\theta \mapsto (X_0, X_1, \cdots, X_n)(\theta)$ is equivalent to requiring that for all $\theta \in \Theta$, the transition function $F$ is a.s. continuously differentiable with respect to $\theta$.

From Equation (5), we can derive the **IPA estimator of** $\nabla \pi_n(f)$ by using a SMC algorithm:

$$I_n^N \triangleq \frac{1}{N} \sum_{i=1}^{N} \left[ f'(x_n^i) z_n^i + f(x_n^i)\big(r_n^i - \frac{1}{N} \sum_{j=1}^{N} r_n^j\big) \right], \tag{6}$$

where $(x_n^i, z_n^i, r_n^i)$ are particles derived by using a SMC algorithm on the augmented Markov chain $(X_t, Z_t, R_t)$ described in Algorithm 2.

---

**Algorithm 2** IPA estimation of $\nabla \pi_n$
<hr>

**for** $t = 1$ **to** $n$ **do**
    For all $i \in \{1, \ldots, N\}$ do
    Sample $u_{t-1}^i \overset{iid}{\sim} \nu$ and set $\tilde{x}_t^i = F(x_{t-1}^i, u_{t-1}^i)$,
    Set $\tilde{z}_t^i = \nabla F(x_{t-1}^i, u_{t-1}^i) + F'(x_{t-1}^i, u_{t-1}^i) z_{t-1}^i$,
    Set $\tilde{r}_t^i = r_{t-1}^i + \frac{G_t'(\tilde{x}_t^i)\tilde{z}_t^i + \nabla G_t(\tilde{x}_t^i)}{G_t(\tilde{x}_t^i)}$, and compute the weights $w_{i,t} = \frac{G_t(\tilde{x}_t^i)}{\sum_j G_t(\tilde{x}_t^j)}$
    Set $(x_t^i, z_t^i, r_t^i) = (\tilde{x}_t^{k_i}, \tilde{z}_t^{k_i}, \tilde{r}_t^{k_i})$, where $k_{1:N}$ are the indices selected from $w_t^{1:N}$,
**end for**
RETURN: $I_n^N = \frac{1}{N} \sum_{i=1}^{N} \left[ f'(x_n^i) z_n^i + f(x_n^i) \left( r_n^i - \frac{1}{N} \sum_{j=1}^{N} r_n^j \right) \right]$

---

**Proposition 2.** *Under the assumptions of Proposition 1, the estimator $I_n^N$ defined by (6) has a bias $O(N^{-1})$ and is consistent with $\nabla \pi_n(f)$, i.e. $\mathbb{E}[I_n^N] = \nabla \pi_n(f) + O(N^{-1})$, and $\lim_{N \to \infty} I_n^N = \nabla \pi_n(f)$ almost surely. In addition, its (asymptotic) variance is $O(N^{-1})$.*

*Proof.* We use the general SMC convergence properties for Feynman-Kac (FK) models (see [Del04] or [DM08]) which, applied to a FK flow with Markov chain $X_{0:n}$, (random) potential functions $G(X_{0:n})$, and test function $H(X_{0:n})$, states that the SMC estimate: $\frac{1}{N} \sum_{i=1}^{N} H(x_{0:n}^i)$ is consistent with $\frac{\mathbb{E}[H(X_{0:n}) \prod_{t=0}^{n} G(X_t)]}{\mathbb{E}[\prod_{t=0}^{n} G(X_t)]}$. Moreover, an asymptotic expression of the bias, given in [DMDP07], shows that it is of order $O(N^{-1})$. Applying those results to the test function $H \triangleq f'(X_n)Z_n + R_n(f(X_n) - \pi_n(f))$, using the representation (5) of the gradient, we deduce that the SMC estimator (6) is asymptotically unbiased and consistent with $\nabla \pi_n(f)$. Now the asymptotic variance is $O(N^{-1})$ since the Central Limit Theorem (see e.g. [Del04, DM08]) applies to the IPA estimator (6) of (5). $\square$

**Remark 1.** *Notice that the computation of the gradient estimator requires $O(nNmd)$ (where $m$ is the dimension of $X$) elementary operations, which is linear in the number of particles $N$ and linear in the number of parameters $d$, and has memory requirement $O(Nmd)$.*

## 4.2 Gradient of the log-likelihood

In the Maximum Likelihood approach for the problem of parameter identification, one may follow a stochastic gradient method for maximizing the log-likelihood $l_n(\theta)$ where the gradient

$$\nabla l_n(\theta) = \sum_{t=0}^{n-1} \frac{\nabla \pi_{t+1|t}(G_{t+1})}{\pi_{t+1|t}(G_{t+1})}$$

is obtained by estimating each term $\nabla \pi_{t+1|t}(G_{t+1})$ of the sum using a similar decomposition as in (5) and (4) for the predictive distribution applied to $G_{t+1}$:

$$
\begin{aligned}
\nabla \pi_{t+1|t}(G_{t+1}) &= \nabla \left[ \frac{\mathbb{E}[G_{t+1}(X_{t+1}) \prod_{k=0}^{t} G_k(X_k)]}{\mathbb{E}[\prod_{k=0}^{t} G_k(X_k)]} \right] \\
&= \frac{\nabla \mathbb{E}[G_{t+1}(X_{t+1}) \prod_{k=0}^{t} G_k(X_k)]}{\mathbb{E}[\prod_{k=0}^{t} G_k(X_k)]} - \pi_{t+1|t}(G_{t+1}) \frac{\nabla \mathbb{E}[\prod_{k=0}^{t} G_k(X_k)]}{\mathbb{E}[\prod_{k=0}^{t} G_k(X_k)]}
\end{aligned}
$$

with

$$
\begin{aligned}
\nabla \mathbb{E}[G_{t+1}(X_{t+1}) \prod_{k=0}^{t} G_k(X_k)] &= \mathbb{E}\Big[ \Big( \nabla G_{t+1}(X_{t+1}) + G'_{t+1}(X_{t+1}) \nabla X_{t+1} \\
&\quad + G_{t+1}(X_{t+1}) \sum_{k=0}^{t} \frac{G'_k(X_k) \nabla X_k + \nabla G_k(X_k)}{G_k(X_k)} \Big) \prod_{k=0}^{t} G_k(X_k) \Big].
\end{aligned}
$$

We deduce the **IPA estimator of** $\nabla l_n(\theta)$

$$
J_n^N \triangleq \sum_{t=1}^{n} \frac{\sum_{i=1}^{N} \left( \nabla G_t(\tilde{x}_t^i) + G'_t(\tilde{x}_t^i) \tilde{z}_t^i + G_t(\tilde{x}_t^i)(r_{t-1}^i - \frac{1}{N} \sum_j r_{t-1}^j) \right)}{\sum_{i=1}^{N} G_t(\tilde{x}_t^i)},
$$

where $(x_n^i, z_n^i, r_n^i)$ (and $(\tilde{x}_n^i, \tilde{z}_n^i, \tilde{r}_n^i)$) are particles derived by using a SMC algorithm on the augmented Markov chain $(X_t, Z_t, R_t)$ described in the previous subsection. Using similar arguments as those detailed in proofs of Propositions 1 and 2, we have that this estimator is asymptotically unbiased and consistent with $\nabla l_n(\theta)$.

The resulting gradient algorithm is described in Algorithm 3. The steps $\gamma_k$ are chosen appropriately so that local convergence occurs (e.g. such that $\sum_{k \geq 1} \gamma_k = \infty$ and $\sum_{k \geq 1} \gamma_k^2 < \infty$), see e.g. [KY97] for a detailed analysis of Stochastic Approximation algorithms.

---

**Algorithm 3** Likelihood Maximization by gradient ascent using the IPA estimator of $\nabla l_n(\theta)$

---

**for** $k = 1, 2, \ldots,$ Number of gradient steps **do**
    Initialize $J_0^N = 0$
    **for** $t = 1$ **to** $n$ **do**
        For all $i \in \{1, \ldots, N\}$ do
        Sample $u_{t-1}^i \overset{iid}{\sim} \nu$ and set $\tilde{x}_t^i = F(x_{t-1}^i, u_{t-1}^i)$,
        Set $\tilde{z}_t^i = \nabla F(x_{t-1}^i, u_{t-1}^i) + F'(x_{t-1}^i, u_{t-1}^i) z_{t-1}^i$,
        Set $J_t^N = J_{t-1}^N + \frac{\sum_{i=1}^{N} \left( \nabla G_t(\tilde{x}_t^i) + G'_t(\tilde{x}_t^i) \tilde{z}_t^i + G_t(\tilde{x}_t^i)(r_{t-1}^i - \frac{1}{N} \sum_j r_{t-1}^j) \right)}{\sum_{i=1}^{N} G_t(\tilde{x}_t^i)}$,
        Set $\tilde{r}_t^i = r_{t-1}^i + \frac{G'_t(\tilde{x}_t^i) \tilde{z}_t^i + \nabla G_t(\tilde{x}_t^i)}{G_t(\tilde{x}_t^i)}$ and compute the weights $w_t^i = \frac{G_t(\tilde{x}_t^i)}{\sum_j G_t(\tilde{x}_t^j)}$
        Set $(x_t^i, z_t^i, r_t^i) = (\tilde{x}_t^{k_i}, \tilde{z}_t^{k_i}, \tilde{r}_t^{k_i})$, where $k_{1:N}$ are indices selected from $w_t^{1:N}$.
    **end for**
    Perform a gradient ascent step: $\theta_k = \theta_{k-1} + \gamma_k J_n^N(\theta_{k-1})$
**end for**

---

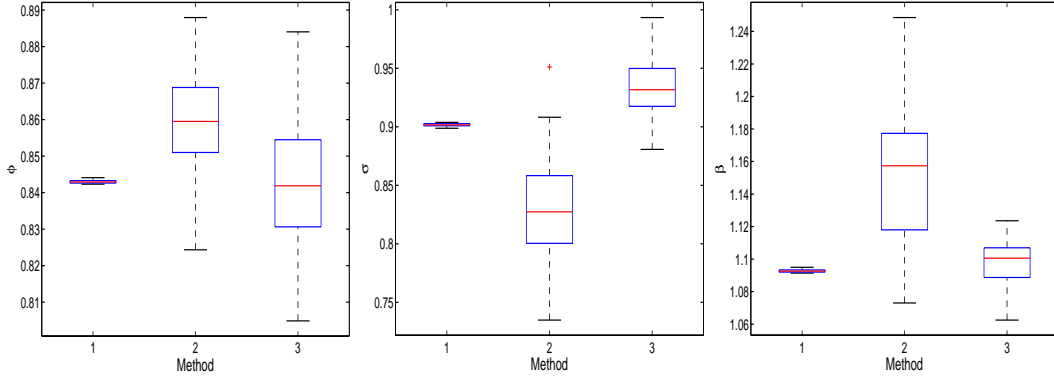

Figure 1: Box-and-whiskers plots of the three parameters $(\phi, \sigma, \beta)$ estimates for the $AR_1$ model with $\theta^\star = (0.8, 1.0, 1.0)$. We compare three methods: (1) Kalman, (2) EM and (3) IPA. Here we used $n = 500$ observations and $N = 10^2$ particles.

## 5  Numerical experiments

We consider two typical problems and report our results focussing on the variance of the estimator:

**Autoregressive model** $AR_1$ is a simple linear-Gaussian HMMs thus may be solved by other methods (such as Kalman filtering and EM algorithms) which enables to compare the performances of several algorithms for parameter identification. The dynamics are

$$X_0 \sim \mathcal{N}(0, \sigma^2), \quad \text{and for } t \geq 1, \quad \begin{aligned} X_t &= \phi X_{t-1} + \sigma U_t, \\ Y_t &= X_t + \beta V_t, \end{aligned} \quad (7)$$

where $U_t \overset{i.i.d.}{\sim} \mathcal{N}(0, 1)$ and $V_t \overset{i.i.d.}{\sim} \mathcal{N}(0, 1)$ are independent sequences of random variables, and $\theta = (\phi, \sigma, \beta)$ is a three-dimensional parameter in $(\mathbb{R}_+)^3$.

**Stochastic volatility model** is very popular in the field of quantitative finance [ME07] to evaluate derivative securities, such as options. This is a non-linear non-Gaussian model, so the Kalman method cannot be used anymore. The dynamics are

$$X_0 \sim \mathcal{N}(0, \sigma^2), \quad \text{and for } t \geq 1, \quad \begin{aligned} X_t &= \phi X_{t-1} + \sigma U_t, \\ Y_t &= \beta \exp(X_t/2)\, V_t, \end{aligned} \quad (8)$$

where again $U_t \overset{i.i.d.}{\sim} \mathcal{N}(0, 1)$ and $V_t \overset{i.i.d.}{\sim} \mathcal{N}(0, 1)$ and the parameter $\theta = (\phi, \sigma, \beta) \in (\mathbb{R}_+)^3$.

### 5.1  Parameter identification

Figure 1 shows the results of our IPA gradient estimator for the $AR_1$ parameter identification problem and compares those with two other methods: Kalman filter (K) and EM (which apply since the model is linear-Gaussian). The unknown parameter used is $\theta^* = (0.8, 1.0, 1.0)$. Notice the apparent bias of the three methods in the estimation of $\theta^*$ (even for Kalman which provides here the exact filtering distribution) since the number of observations $n = 500$ is finite. For IPA, we used $N = 10^2$ particles and 150 gradient iterations. Algorithm 3 was run 50 times with random starting points uniformly drawn between $[\underline{\theta}, \bar{\theta}]$, where $\underline{\theta} = (0.5, 0.5, 0.5)$ and $\bar{\theta} = (1.0, 1.5, 1.5)$ in order to illustrate that the method is not sensitive to the starting point.

We observe that in terms of estimation accuracy, IPA is very competitive to the other methods, Kalman and EM, which are designed for specific models (here linear-Gaussian). The IPA method applies to general models, for example, to the stochastic volatility model. Figure 2 shows the sets of estimates of $\theta^\star = (0.8, 1.0, 1.0)$ using IPA with $n = 10^3$ observations and $N = 10^2$ particles (no comparison is made here since Kalman does not apply and EM becomes more complicated).

### 5.2  Variance study for Score and IPA algorithms

IPA and Score methods provide gradient estimators for general models. We compare the variance of the corresponding estimators of the gradient $\nabla l_n$ for the $AR_1$ since for this model we know its exact value (using Kalman).

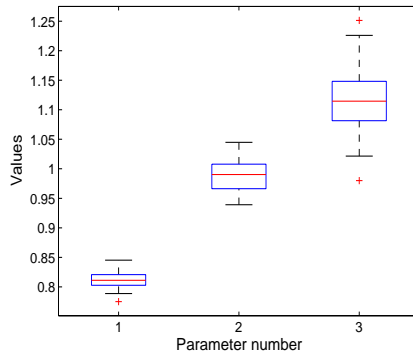

Figure 2: Box-and-whiskers plots of the three parameters $(\phi, \sigma, \beta)$ estimates for the IPA method applied to the stochastic volatility model with $\theta^\star = (0.8, 1.0, 1.0)$. We used $n = 10^3$ observations and $N = 10^2$ particles.

Figure 3 shows the variance of the IPA and Score estimators of the partial derivative $\partial_\sigma l_n$ (we focused our study on $\sigma$ since the problem of volatility estimation is challenging, and also because the value of $\sigma$ influences the respective performances of the two algorithms, which is not the case for the other parameters $\phi, \beta$). We used $n = N = 10^3$. The IPA estimator performs better than the Score estimator for small values of $\sigma$. On the other hand, in case of huge variance in the state model, it is better to use the Score estimator.

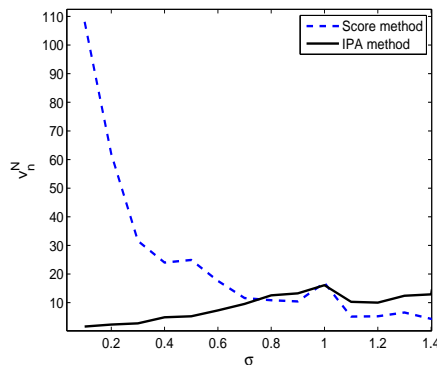

Figure 3: Variance of the log-likelihood derivative $\partial_\sigma l_n$ computed with both the IPA and Score methods. The true parameter is $\theta^* = (\phi^\star, \sigma^\star, \beta^\star) = (0.8, 1.0, 1.0)$ and the estimations are computed at $\theta = (0.7, \sigma, 0.9)$.

Let us mention that the variance of the IPA (as well as Score) estimator increases when the number of observations $n$ increases. However, under weak conditions on the HMM [LM00], the filtering distribution and its gradient forget exponentially fast the initial distribution. This property has already been used for EM estimators in [CM05] to show that fixed-lag smoothing drastically reduces the variance without significantly raising the bias. Similar smoothing (either fixed-lag or discounted) would provide efficient variance reduction techniques for the IPA estimator as well.

## 6 Conclusions

We proposed a sensitivity analysis in HMMs based on an Infinitesimal Perturbation Analysis and provided a computationally efficient gradient estimator that provides an interesting alternative to the usual Score method. We showed how this analysis may be used for estimating the gradient of the log-likelihood in a gradient-based likelihood maximization approach for the purpose of parameter identification. Finally let us mention that estimators of higher-order derivatives (e.g. Hessian) could be derived as well along this IPA approach, which would enable to use more sophisticated optimization techniques (e.g. Newton method).

# References

[CDM08]    P.A. Coquelin, R. Deguest, and R. Munos. Particle filter-based policy gradient in POMDPs. In *Neural Information Processing Systems*, 2008.

[CGN01]    F. Cérou, F. Le Gland, and N. J. Newton. Stochastic particle methods for linear tangent filtering equations. In J.-L. Menaldi, E. Rofman, and A. Sulem, editors, *Optimal Control and PDE's - Innovations and Applications, in honor of Alain Bensoussan's 60th anniversary*, pages 231–240. IOS Press, 2001.

[CM05]    O. Cappé and E. Moulines. On the use of particle filtering for maximum likelihood parameter estimation. *European Signal Processing Conference*, 2005.

[Del04]    P. Del Moral. *Feynman-Kac Formulae, Genealogical and Interacting Particle Systems with Applications*. Springer, 2004.

[DFG01]    A. Doucet, N. De Freitas, and N. Gordon. *Sequential Monte Carlo Methods in Practice*. Springer, 2001.

[DM01]    R. Douc and C. Matias. Asymptotics of the maximum likelihood estimator for general hidden markov models. *Bernouilli*, 7:381–420, 2001.

[DM08]    R. Douc and E. Moulines. Limit theorems for weighted samples with applications to sequential monte carlo methods. *Annals of Statistics*, 36:5:2344–2376, 2008.

[DMDP07]    P. Del Moral, A. Doucet, and GW Peters. Sharp Propagation of Chaos Estimates for Feynman–Kac Particle Models. *SIAM Theory of Probability and its Applications*, 51 (3):459–485, 2007.

[DT03]    A. Doucet and V.B. Tadic. Parameter estimation in general state-space models using particle methods. *Ann. Inst. Stat. Math*, 2003.

[FLM03]    J. Fichoud, F. LeGland, and L. Mevel. Particle-based methods for parameter estimation and tracking : numerical experiments. Technical Report 1604, IRISA, 2003.

[Gla91]    P. Glasserman. *Gradient estimation via perturbation analysis*. Kluwer, 1991.

[GSS93]    N. Gordon, D. Salmond, and A. F. M. Smith. Novel approach to nonlinear and non-gaussian bayesian state estimation. In *Proceedings IEE-F*, volume 140, pages 107–113, 1993.

[Kit96]    G. Kitagawa. Monte-Carlo filter and smoother for non-Gaussian nonlinear state space models. *J. Comput. Graph. Stat.*, 5:1–25, 1996.

[KY97]    H. J. Kushner and G. Yin. *Stochastic Approximation Algorithms and Applications*. Springer-Verlag, Berlin and New York, 1997.

[LM00]    F. LeGland and L. Mevel. Exponential forgetting and geometric ergodicity in hidden markov models. *mathematic and control sugnal and systems*, 13:63–93, 2000.

[ME07]    R. Mamon and R.J. Elliott. Hidden markov models in finance. *International Series in Operations Research and Management Science*, 104, 2007.

[Pap07]    A. Papavasiliou. A uniformly convergent adaptive particle filter. *Journal of Applied Probability*, 42 (4):1053–1068, 2007.

[PDS05]    G. Poyadjis, A. Doucet, and S.S. Singh. Particle methods for optimal filter derivative: Application to parameter estimation. In *IEEE ICASSP*, 2005.

[Pfl96]    G. Pflug. *Optimization of Stochastic Models: The Interface Between Simulation and Optimization*. Kluwer Academic Publishers, 1996.

[Poy06]    G. Poyiadjis. *Particle Method for Parameter Estimation in General State Space Models*. PhD thesis, University of Cambridge, 2006.

[Sto02]    G. Storvik. Particle filters for state-space models with the presence of unknown static parameters. *IEEE Transactions on Signal Processing*, 50:281–289, 2002.

